# Interpreting images by propagating Bayesian beliefs

**Yair Weiss**
Dept. of Brain and Cognitive Sciences
Massachusetts Institute of Technology
E10-120, Cambridge, MA 02139, USA
yweiss@psyche.mit.edu

## Abstract

A central theme of computational vision research has been the realization that reliable estimation of local scene properties requires propagating measurements across the image. Many authors have therefore suggested solving vision problems using architectures of locally connected units updating their activity in parallel. Unfortunately, the convergence of traditional relaxation methods on such architectures has proven to be excruciatingly slow and in general they do not guarantee that the stable point will be a global minimum.

In this paper we show that an architecture in which *Bayesian Beliefs* about image properties are propagated between neighboring units yields convergence times which are several orders of magnitude faster than traditional methods and avoids local minima. In particular our architecture is non-iterative in the sense of Marr [5]: at every time step, the local estimates at a given location are optimal given the information which has already been propagated to that location. We illustrate the algorithm's performance on real images and compare it to several existing methods.

## 1 Theory

The essence of our approach is shown in figure 1. Figure 1a shows the prototypical ill-posed problem: interpolation of a function from sparse data. Figure 1b shows a traditional relaxation approach to the problem: a dense array of units represents the value of the interpolated function at discretely sampled points. The activity of a unit is updated based on the local data (in those points where data is available) and the activity of the neighboring points. As discussed below, the local update rule can

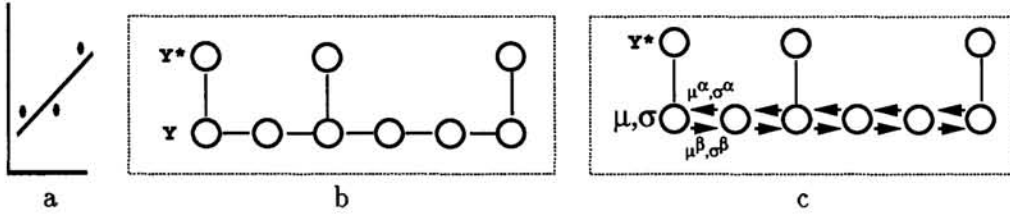

Figure 1: **a.** a prototypical ill-posed problem **b.** Traditional relaxation approach: dense array of units represent the value of the interpolated function. Units update their activity based on local information and the activity of neighboring units. **c.** The Bayesian Belief Propagation (BBP) approach. Units transmit probabilities and combine them according to probability calculus in two non-interacting streams.

be defined such that the network converges to a state in which the activity of each unit corresponds to the value of the globally optimal interpolating function. Figure 1c shows the Bayesian Belief Propagation (BBP) approach to the problem. As in the traditional approach the function is represented by the activity of a dense array of units. However the units transmit probabilities rather than single estimates to their neighbors and combine the probabilities according to the probability calculus.

To formalize the above discussion, let $y_k$ represent the activity of a unit at location $k$, and let $y_k^*$ be noisy samples from the true function. A typical interpolation problem would be to minimize:

$$J(Y) = \sum_k w_k(y_k - y_k^*)^2 + \lambda \sum_i (y_i - y_{i+1})^2 \tag{1}$$

Where we have defined $w_k = 0$ for grid points with no data, and $w_k = 1$ for points with data. Since $J$ is quadratic, any local update in the direction of the gradient will converge to the optimal estimate. This yields updates of the sort:

$$y_k \leftarrow y_k + \eta_k(\lambda(\frac{y_{k-1} + y_{k+1}}{2} - y_k) + w_k(y_k^* - y_k)) \tag{2}$$

Relaxation algorithms differ in their choice of $\eta$: $\eta = 1/(\lambda + w_k)$ corresponds to Gauss-Seidel relaxation and $\eta = 1.9/(\lambda + w_k)$ corresponds to successive over relaxation (SOR) which is the method of choice for such problems [10].

To derive a BBP update rule for this problem, note that that minimizing $J(Y)$ is equivalent to maximizing the posterior probability of $Y$ given $Y^*$ assuming the following generative model:

$$y_{i+1} = y_i + \nu \tag{3}$$
$$y_i^* = w_i y_i + \eta \tag{4}$$

Where $\nu \sim N(0, \sigma_R)$, $\eta \sim N(0, \sigma_D)$. The ratio of $\sigma_D$ to $\sigma_R$ plays a role similar to that of $\lambda$ in the original cost functional.

The advantage of considering the cost functional as a posterior is that it enables us to use the methods of Hidden Markov Models, Bayesian Belief Nets and Optimal Estimation to derive local update rules (cf. [6, 7, 1]). Denote the posterior by $P_i(u) = P(Y_i = u|Y^*)$, the Markovian property allows us to factor $P_i(u)$ into three terms: one depending on the local data, another depending on data to the left of $i$ and a third depending on data to the right of $i$. Thus:

$$P_i(u) = c\alpha_i(u)L_i(u)\beta_i(u) \tag{5}$$

where $\alpha_i(u) = P(Y_i = u|Y_{1,i-1}^*)$, $\beta_i(u) = P(Y_i = u|Y_{i+1,N}^*)$, $L_i(u) = P(Y_i^*|Y_i = u)$ and $c$ denotes a normalizing constant. Now, denoting the conditional $C_i(u, v) =$

$P(Y_i = u | Y_{i-1} = v)$, $\alpha_i(u)$ can be written in terms of $\alpha_{i-1}(v)$:

$$\alpha_i(u) = c \int_v \alpha_{i-1}(v) C_i(u,v) L_{i-1}(v) \tag{6}$$

where c denotes another normalizing constant. A symmetric equation can be written for $\beta_i(u)$.

This suggests a propagation scheme where units represent the probabilities given in the left hand side of equations 5–6 and updates are based on the right hand side, i.e. on the activities of neighboring units. Specifically, for a Gaussian generating process the probabilities can be represented by their mean and variance. Thus denote $P_i \sim N(\mu_i, \sigma_i)$, and similarly $\alpha_i \sim N(\mu_i^\alpha, \sigma_i^\alpha)$ and $\beta_i \sim N(\mu_i^\beta, \sigma_i^\beta)$. Performing the integration in 6 gives a Kalman-Filter like update for the parameters:

$$\mu_i \leftarrow \frac{\frac{w_i}{\sigma_D} Y_i^* + \frac{1}{\sigma_i^\alpha} \mu_i^\alpha + \frac{1}{\sigma_i^\beta} \mu_i^\beta}{\frac{w_i}{\sigma_D} + \frac{1}{\sigma_i^\alpha} + \frac{1}{\sigma_i^\beta}} \tag{7}$$

$$\mu_i^\alpha \leftarrow \frac{\frac{1}{\sigma_{i-1}^\alpha} \mu_{i-1}^\alpha + \frac{w_{i-1}}{\sigma_D} Y_{i-1}^*}{\frac{1}{\sigma_{i-1}^\alpha} + \frac{w_{i-1}}{\sigma_D}} \tag{8}$$

$$\sigma_i^\alpha \leftarrow \sigma_R + (\frac{1}{\sigma_{i-1}^\alpha} + \frac{w_{i-1}}{\sigma_D})^{-1} \tag{9}$$

(the update rules for the parameters of $\beta$ are analogous)

So far we have considered continuous estimation problems but identical issues arise in labeling problems, where the task is to estimate a label $L_k$ which can take on $M$ discrete values. We will denote $L_k(m) = 1$ if the label takes on value $m$ and zero otherwise. Typically one minimizes functionals of the form:

$$J(L) = \sum_m \sum_k V_k(m) L_k(m) - \lambda \sum_m \sum_k L_k(m) L_{k+1}(m) \tag{10}$$

Traditional relaxation labeling algorithms minimize this cost functional with updates of the form:

$$L_k \leftarrow f(V_k, L_{k-1}, L_k, L_{k+1}) \tag{11}$$

Again different relaxation labeling algorithms differ in their choice of $f$. A linear sum followed by a threshold gives the discrete Hopfield network updates, a linear sum followed by a "soft" threshold gives the continuous or mean-field Hopfield updates and yet another form gives the relaxation labeling algorithm of Rosenfeld et al. (see [3] for a review of relaxation labeling methods ).

To derive a BBP algorithm for this case one can again rewrite $J$ as the posterior of a Markov generating process, and calculate $P(L_k(m) = 1)$ for this process.[1]. This gives the same expressions as in equations 5–6 with the integral replaced by a linear sum. Since the probabilities here are not Gaussian, the $\alpha_i, \beta_i, P_i$ will not be represented by their mean and variances, but rather by a vector of length $M$. Thus the update rule for $\alpha_i$ will be:

$$\alpha_i(k) \leftarrow c \sum_l \alpha_{i-1}(l) C_i(k,l) L_{i-1}(l) \tag{12}$$

(and similarly for $\beta$.)

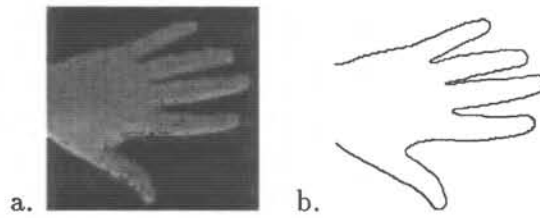

Figure 2: **a.** the first frame of a sequence. The hand is translated to the left. **b.** contour extracted using standard methods

## 1.1 Convergence

Equations 5–6 are mathematical identities. Hence, it is possible to show [6] that after $N$ iterations the activity of units $P_i$ will converge to the correct posteriors, where $N$ is the maximal distance between any two units in the architecture, and an iteration refers to one update of all units. Furthermore, we have been able to show that after $n < N$ iterations, the activity of unit $P_i$ is guaranteed to represent the probability of the hidden state at location $i$ given all data within distance $n$.

This guarantee is significant in the light of a distinction made by Marr (1982) regarding local propagation rules. In a scheme where units only communicate with their neighbors, there is an obvious limit on how fast the information can reach a given unit: i.e. after $n$ iterations the unit can only know about information within distance $n$. Thus there is a minimal number of iterations required for all data to reach all units. Marr distinguished between two types of iterations – those that are needed to allow the information to reach the units, versus those that are used to refine an estimate based on information that has already arrived. The significance of the guarantee on $P_i$ is that it shows that BBP only uses the first type of iteration – iterations are used only to allow more information to reach the units. Once the information has arrived, $P_i$ represents the correct posterior given that information and no further iterations are needed to refine the estimate. Moreover, we have been able to show that propagations schemes that do not propagate probabilities (such as those in equations 2) will in general *not* represent the optimal estimate given information that has already arrived.

To summarize, both traditional relaxation updates as in equation 2 and BBP updates as in equations 7–9 give simple rules for updating a unit's activity based on local data and activities of neighboring units. However, the fact that BBP updates are based on the probability calculus guarantees that a unit's activity will be optimal given information that has already arrived and gives rise to a qualitative difference between the convergence of these two types of schemes. In the next section, we will demonstrate this difference in image interpretation problems.

## 2 Results

Figure 2a shows the first frame of a sequence in which the hand is translated to the left. Figure 2b shows the bounding contour of the hand extracted using standard techniques.

### 2.1 Motion propagation along contours

Local measurements along the contour are insufficient to determine the motion. Hildreth [2] suggested to overcome the local ambiguity by minimizing the following

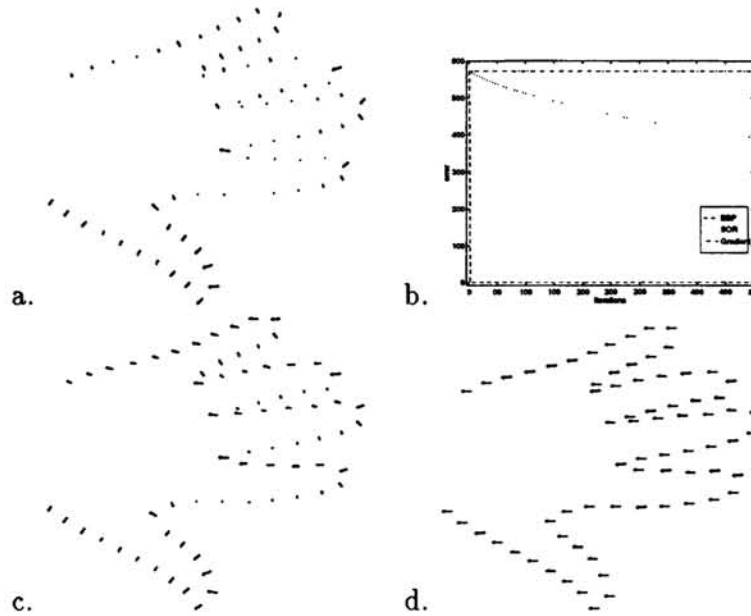

Figure 3: **a.** Local estimate of velocity along the contour. **b.** Performance of SOR, gradient descent and BBP as a function of time. BBP converges orders of magnitude faster than SOR. **c.** Motion estimate of SOR after 500 iterations. **d.** Motion estimate of BBP after 3 iterations.

cost functional:

$$J(V) = \sum_k (dx_k^t v_k + dt_k)^2 + \lambda \sum_k \|v_{k+1} - v_k\|^2 \qquad (13)$$

where $dx, dt$ denote the spatial and temporal image derivatives and $v_k$ denotes the velocity at point $k$ along the contour. This functional is analogous to the interpolation functional (eq. 1) and the derivation of the relaxation and BBP updates are also analogous.

Figure 3a shows the estimate of motion based solely on local information. The estimates are wrong due to the aperture problem. Figure 3b shows the performance of three propagation schemes: gradient descent, SOR and BBP. Gradient descent converges so slowly that the improvement in its estimate can not be discerned in the plot. SOR converges much faster than gradient descent but still has significant error after 500 iterations. BBP gets the correct estimate after 3 iterations ! (Here and in all subsequent plots an iteration refers to one update of all units in the network). This is due to the fact that after 3 iterations, the estimate at location $k$ is the optimal one given data in the interval $[k - 3, k + 3]$. In this case, there is enough data in every such interval along the contour to correctly estimate the motion. Figure 3c shows the estimate produced by SOR after 500 iterations. Even with simple visual inspection it is evident that the estimate is quite wrong. Figure 3d shows the (correct) estimate produced by BBP after 3 iterations.

## 2.2 Direction of figure propagation

The extracted contour in figure 2 bounds a dark and a light region. *Direction of figure* (DOF) (e.g. [9]) refers to which of these two regions is figure and which is ground. A local cue for DOF is convexity - given three neighboring points along the contour we prefer the DOF that makes the angle defined by those points acute

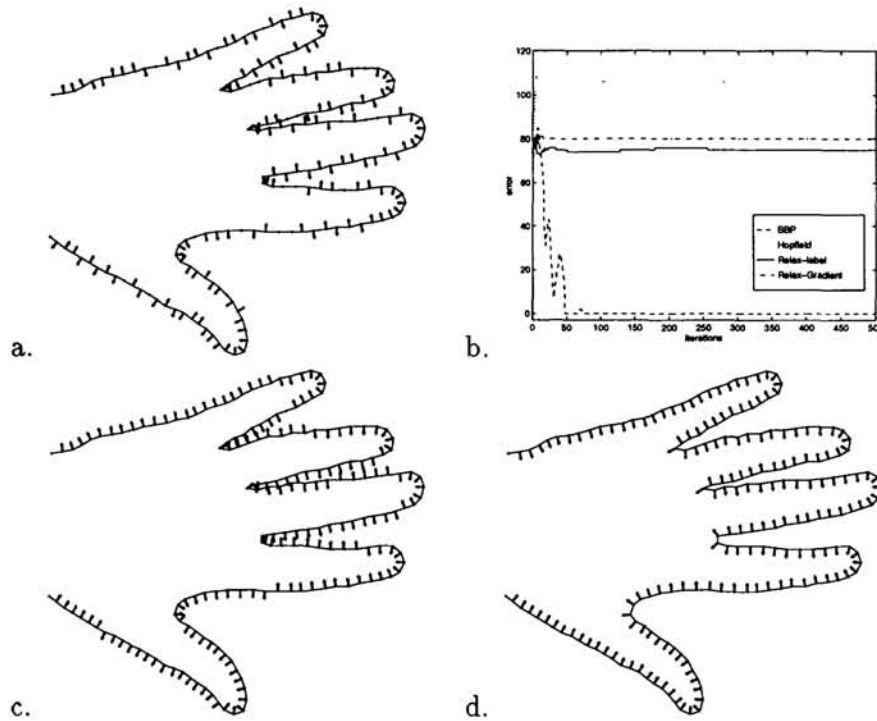

Figure 4: **a.** Local estimate of DOF along the contour. **b.** Performance of Hopfield, gradient descent, relaxation labeling and BBP as a function of time. BBP is the only method that converges to the global minimum. **c.** DOF estimate of Hopfield net after convergence. **d.** DOF estimate of BBP after convergence.

rather than obtuse. Figure 4a shows the results of using this local cue on the hand contour. The local cue is not sufficient.

We can overcome the local ambiguity by minimizing a cost functional that takes into account the DOF at neighboring points in addition to the local convexity. Denote by $L_k(m)$ the DOF at point $k$ along the contour and define

$$J(L) = \sum_m \sum_k V_k(m) L_k(m) - \lambda \sum_m \sum_k L_k(m) L_{k+1}(m) \tag{14}$$

with $V_k(m)$ determined by the acuteness of the angle at location $k$.

Figure 4b shows the performance of four propagation algorithms on this task: three traditional relaxation labeling algorithms (MF Hopfield, Rosenfeld et al, constrained gradient descent) and BBP. All three traditional algorithms converge to a local minimum, while the BBP converges to the global minimum. Figure 4c shows the local minimum reached by the Hopfield network and figure 4d shows the correct solution reached by the BBP algorithm. Recall (section 1.1) that BBP is guaranteed to converge to the correct posterior given all the data.

## 2.3 Extensions to 2D

In the previous two examples ambiguity was reduced by combining information from other points on the same contour. There exist, however, cases when information should be propagated to all points in the image. Unfortunately, such propagation problems correspond to Markov Random Field (MRF) generative models, for which calculation of the posterior cannot be done efficiently. However, Willsky and his

colleagues [4] have recently shown that MRFs can be approximated with hierarchical or multi-resolution models. In current work, we have been using the multi-resolution generative model to derive local BBP rules. In this case, the Bayesian beliefs are propagated between neighboring units in a pyramidal representation of the image. Although this work is still in preliminary stages, we find encouraging results in comparison with traditional 2D relaxation schemes.

## 3   Discussion

The update rules in equations 5–6 differ slightly from those derived by Pearl [6] in that the quantities $\alpha, \beta$ are conditional probabilities and hence are constantly normalized to sum to unity. Using Pearl's original algorithm for sequences as long as the ones we are considering will lead to messages that become vanishingly small. Likewise our update rules differ slightly from the forward-backward algorithm for HMMs [7] in that ours are based on the assumption that all states are equally likely apriore and hence the updates are symmetric in $\alpha$ and $\beta$. Finally, equation 9 can be seen as a variant of a Riccati equation [1].

In addition to these minor notational differences, the context in which we use the update rules is different. While in HMMs and Kalman Filters, the updates are seen as interim calculations toward calculating the posterior, we use these updates in a parallel network of local units and are interested in how the estimates of units in this network improve as a function of iteration. We have shown that an architecture that propagates Bayesian beliefs according to the probability calculus yields orders of magnitude improvements in convergence over traditional schemes that do not propagate probabilities. Thus image interpretation provides an important example of a task where it pays to be a Bayesian.

### Acknowledgments

I thank E. Adelson, P. Dayan, J. Tenenbaum and G. Galperin for comments on versions of this manuscript; M.I. Jordan for stimulating discussions and for introducing me to Bayesian nets. Supported by a training grant from NIGMS.

## Footnotes

[1]For certain special cases, knowing $P(L_k(m) = 1)$ is not sufficient for choosing the sequence of labels that minimizes $J$. In those cases one should do belief revision rather than propagation [6]

## References

[1] Arthur Gelb, editor. *Applied Optimal Estimation.* MIT Press, 1974.

[2] E. C. Hildreth. *The Measurement of Visual Motion.* MIT Press, 1983.

[3] S.Z. Li. *Markov Random Field Modeling in Computer Vision.* Springer-Verlag, 1995.

[4] Mark R. Luettgen, W. Clem Karl, and Allan S. Willsky. Efficient multiscale regularization with application to the computation of optical flow. *IEEE Transactions on image processing,* 3(1):41–64, 1994.

[5] D. Marr. *Vision.* H. Freeman and Co., 1982.

[6] Judea Pearl. *Probabilistic Reasoning in Intelligent Systems: Networks of Plausible Inference.* Morgan Kaufmann, 1988.

[7] Lawrence Rabiner and Biing-Hwang Juang. *Fundamentals of Speech recognition.* PTR Prentice Hall, 1993.

[8] A. Rosenfeld, R. Hummel, and S. Zucker. Scene labeling by relaxation operations. *IEEE Transactions on Systems, Man and Cybernetics,* 6:420–433, 1976.

[9] P. Sajda and L. H. Finkel. Intermediate-level visual representations and the construction of surface perception. *Journal of Cognitive Neuroscience,* 1994.

[10] Gilbert Strang. *Introduction to Applied Mathematics.* Wellesley-Cambridge, 1986.